# JPMAX: Learning to Recognize Moving Objects as a Model-fitting Problem

**Suzanna Becker**
Department of Psychology, McMaster University
Hamilton, Ont. L8S 4K1

## Abstract

Unsupervised learning procedures have been successful at low-level feature extraction and preprocessing of raw sensor data. So far, however, they have had limited success in learning higher-order representations, e.g., of objects in visual images. A promising approach is to maximize some measure of agreement between the outputs of two groups of units which receive inputs physically separated in space, time or modality, as in (Becker and Hinton, 1992; Becker, 1993; de Sa, 1993). Using the same approach, a much simpler learning procedure is proposed here which discovers features in a single-layer network consisting of several populations of units, and can be applied to multi-layer networks trained one layer at a time. When trained with this algorithm on image sequences of moving geometric objects a two-layer network can learn to perform accurate position-invariant object classification.

## 1 LEARNING COHERENT CLASSIFICATIONS

A powerful constraint in sensory data is coherence over time, in space, and across different sensory modalities. An unsupervised learning procedure which can capitalize on these constraints may be able to explain much of perceptual self-organization in the mammalian brain. The problem is to derive an appropriate cost function for unsupervised learning which will capture coherence constraints in sensory signals; we would also like it to be applicable to multi-layer nets to train hidden as well as output layers. Our ultimate goal is for the network to discover natural object classes based on these coherence assumptions.

## 1.1  PREVIOUS WORK

Successive images in continuous visual input are usually views of the same object; thus, although the image pixels may change considerably from frame to frame, the image usually can be described by a small set of consistent object descriptors, or lower-level feature descriptors. We refer to this type of continuity as *temporal coherence*. This sort of structure is ubiquitous in sensory signals, from vision as well as other senses, and can be used by a neural network to derive temporally coherent classifications. This idea has been used, for example, in temporal versions of the Hebbian learning rule to associate items over time (Weinshall, Edelman and Bülthoff, 1990; Földiák, 1991). To capitalize on temporal coherence for higher-order feature extraction and classification, we need a more powerful learning principle.

A promising approach is to maximize some measure of agreement between the outputs of two groups of units which receive inputs physically separated in space, time or modality, as in (Becker and Hinton, 1992; Becker, 1993; de Sa, 1993). This forces the units to extract features which are coherent across the different input sources. Becker and Hinton's (1992) Imax algorithm maximizes the mutual information between the outputs of two modules, $\vec{y_a}$ and $\vec{y_b}$, connected to different parts of the input, $a$ and $b$. Becker (1993) extended this idea to the problem of classifying temporally varying patterns by applying the discrete case of the mutual information cost function to the outputs of a single module at successive time steps, $\vec{y_a}(t)$ and $\vec{y_a}(t+1)$. However, the success of this method relied upon the back-propagation of derivatives to train the hidden layer and it was found to be extremely susceptible to local optima. de Sa's method (1993) is closely related, and minimizes the probability of disagreement between output classifications, $\vec{y_a}(t)$ and $\vec{y_b}(t)$, produced by two modules having different inputs, e.g., from different sensory modalities. The success of this method hinges upon bootstrapping the first layer by initializing the weights to randomly selected training patterns, so this method too is susceptible to the problem of local optima. If we had a more flexible cost function that could be applied to a multi-layer network, first to each hidden layer in turn, and finally to the output layer for classification, so that the two layers could discover genuinely different structure, we might be able to overcome the problem of getting trapped in local optima, yielding a more powerful and efficient learning procedure.

We can analyze the optimal solutions for both de Sa's and Becker's cost functions (see Figure 1 a) and see that both cost functions are maximized by having perfect one-to-one agreement between the two groups of units over all cases, using a one-of-$n$ encoding, i.e., having only a single output unit on for each case. A major limitation of these methods is that they strive for perfect classifications by the units. While this is desirable at the top layer of a network, it is an unsuitable goal for training intermediate layers to detect low-level features. For example, features like oriented edges would not be perfect predictors across spatially or temporally nearby image patches in images of translating and rotating objects. Instead, we might expect that an oriented edge at one location would predict a small range of similar orientations at nearby locations. So we would prefer a cost function whose optimal solution was more like those shown in Figure 1 b) or c). This would allow a feature $i$ in group $a$ to agree with any of several nearby features, e.g. $i-1$, $i$, or $i+1$ in group $b$.

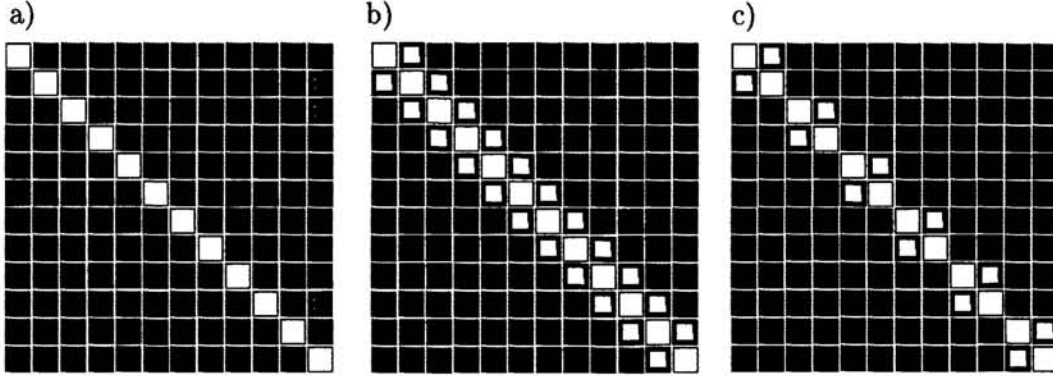

Figure 1: *Three possible joint distributions for the probability that the ith and jth units in two sets of m classification units are both on. White is high density, black is low density. The optimal joint distribution for Becker's and de Sa's algorithms is a matrix with all its density either in the diagonal as in a), or any subset of the diagonal entries for de Sa's method, or a permutation of the diagonal matrix for Becker's algorithm. Alternative distributions are shown in b) and c).*

## 1.2 THE JPMAX ALGORITHM

One way to achieve an arbitrary configuration of agreement over time between two groups of units (as in Figure 1 b) or c)) is to treat the desired configuration as a prior joint probability distribution over their outputs. We can obtain the actual distribution by observing the temporal correlations between pairs of units' outputs in the two groups over an ensemble of patterns. We can then optimize the actual distribution to fit the prior. We now derive two different cost functions which achieve this result. Interestingly, they result in very similar learning rules.

Suppose we have two groups of $m$ units as shown in Figure 2 a), receiving inputs, $\vec{x_a}$ and $\vec{x_b}$, from the same or nearby parts of the input image. Let $C_a(t)$ and $C_b(t)$ be the classifications of the two input patches produced by the network at time step $t$; the outputs of the two groups of units, $\vec{y_a}(t)$ and $\vec{y_b}(t)$, represent these classification probabilities:

$$
\begin{aligned}
y_{ai}(t) &= P(C_a(t) = i) = \frac{e^{net_{ai}(t)}}{\sum_j e^{net_{aj}(t)}} \\
y_{bi}(t) &= P(C_b(t) = i) = \frac{e^{net_{bi}(t)}}{\sum_j e^{net_{bj}(t)}}
\end{aligned}
\tag{1}
$$

(the usual "softmax" output function) where $net_{ai}(t)$ and $net_{bj}(t)$ are the weighted net inputs to units. We could now observe the expected joint probability distribution $q_{ij} = E\left[y_{ai}(t)y_{bj}(t+1)\right]_t = E\left[P(C_a(t) = i, C_b(t+1) = j)\right]_t$ by computing the temporal covariances between the classification probabilities, averaged over the ensemble of training patterns; this joint probability is an $m^2$-valued random variable.

Given the above statistics, one possible cost function we could minimize is the $-$ log probability of the observed temporal covariance between the two sets of units' outputs under some prior distribution (e.g. Figure 1 b) or c)). If we knew the actual *frequency counts* for each (joint) classification $\vec{k} = k_{11}, \ldots, k_{1m}, k_{21}, \ldots, k_{mm}$,

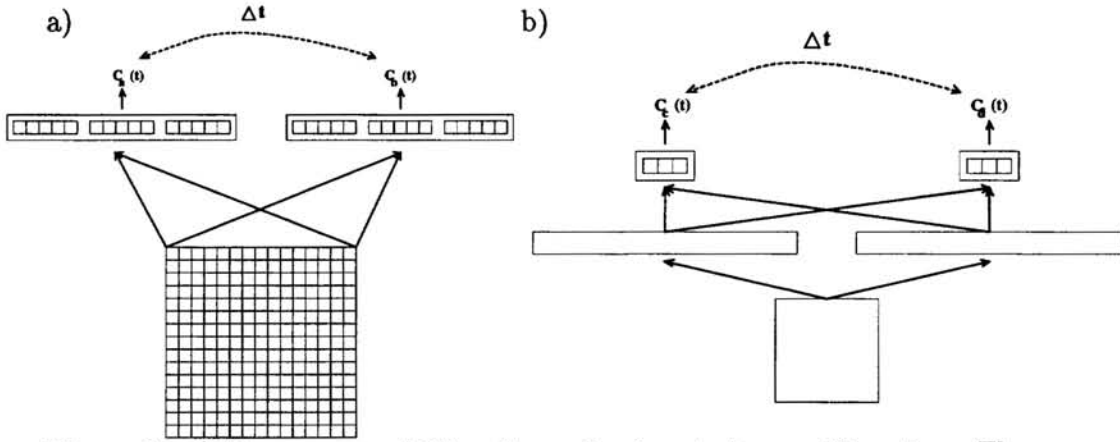

Figure 2: *a) Two groups of 15 units receive inputs from a 2D retina. The groups are able to observe each other's outputs across lateral links with unit time delays. b) A second layer of two groups of 3 units is added to the architecture in a).*

rather than just the observed joint probabilities, $q_{ij} = E\left[\frac{k_{ij}}{n}\right]$, then given our prior model, $p_{11}, \ldots, p_{mm}$, we could compute the probability of the observations under a multinomial distribution:

$$P(\vec{k}) = \frac{n!}{\prod_{i,j} k_{ij}!} \prod_{i,j} p_{ij}^{k_{ij}} \qquad (2)$$

Using the de Moivre-Laplace approximation leads to the following:

$$P(\vec{k}) \approx \frac{1}{\sqrt{(2\pi n)^{m^2-1} \prod_{i,j} p_{ij}}} \exp\left(-\frac{1}{2} \sum_{i,j} \frac{(k_{ij} - np_{ij})^2}{np_{ij}}\right) \qquad (3)$$

Taking the derivative of the - log probability *wrt* $k_{ij}$ leads to a very simple learning rule which depends only on the observed probabilities $q_{ij}$ and priors $p_{ij}$:

$$\frac{\partial - \log P(\vec{k})}{\partial k_{ij}} = \frac{np_{ij} - k_{ij}}{np_{ij}}$$

$$\approx \frac{p_{ij} - q_{ij}}{p_{ij}} \qquad (4)$$

To obtain the final weight update rule, we just multiply this by $n\frac{\partial q_{ij}}{\partial w_{kl}}$. One problem with the above formulation is that the priors $p_{ij}$ must not be too close to zero for the de Moivre-Laplace approximation to hold. In practice, this cost function works well if we simply ignore the derivative terms where the priors are zero.

An alternative cost function (as suggested by Peter Dayan) which works equally well is the Kullback-Liebler divergence or G-error between the desired joint probabilities $p_{ij}$ and the observed probabilities $q_{ij}$:

$$G(p, q) = -\sum_i \sum_j (p_{ij} \log p_{ij} - p_{ij} \log q_{ij}) \qquad (5)$$

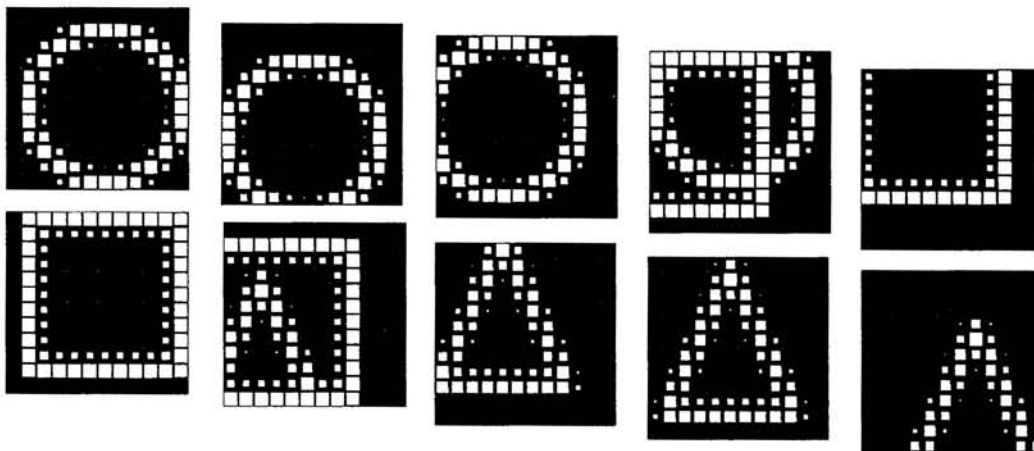

Figure 3: *10 of the 1500 training patterns: geometric objects centered in 36 possible locations on a 12-by-12 pixel grid. Object location varied randomly between patterns.*

The derivative of $G$ *wrt* $q_{ij}$ is:

$$\frac{\partial G}{\partial q_{ij}} = \frac{p_{ij}}{q_{ij}} \tag{6}$$

subject to $\sum_{ij} q_{ij} = 1$ (enforced by the softmax output function). Note the similarity between the learning rules given by equations 4 and 6.

## 2 EXPERIMENTS

The network shown in Figure 2 a) was trained to minimize equation 5 on an ensemble of pattern trajectories of circles, squares and triangles (see Figure 3) for five runs starting from different random initial weights, using a gradient-based learning method. For ten successive frames, the same object would appear, but with the centre varying randomly within the central six-by-six patch of pixels. In the last two frames, another randomly selected object would appear, so that object trajectories overlapped by two frames. These images are meant to be a crude approximation to what a moving observer would see while watching multiple moving objects in a scene; at any given time a single object would be approximately centered on the retina but its exact location would always be jittering from moment to moment.

In these simulations, the prior distribution for the temporal covariances between the two groups of units' outputs was a block-diagonal configuration as in Figure 1 c), but with three five-by-five blocks along the diagonal. Our choice of a block-diagonal prior distribution with three blocks encodes the constraint that units in a given block in one group should try to agree with units in the same block in the other group; so each group should discover three classes of features. The number of units within a block was varied in preliminary experiments, and five units was found to be a reasonable number to capture all instances of each object class (although the performance of the algorithm seemed to be robust with respect to the number of units per block). The learning took about 3000 iterations of steepest descent with

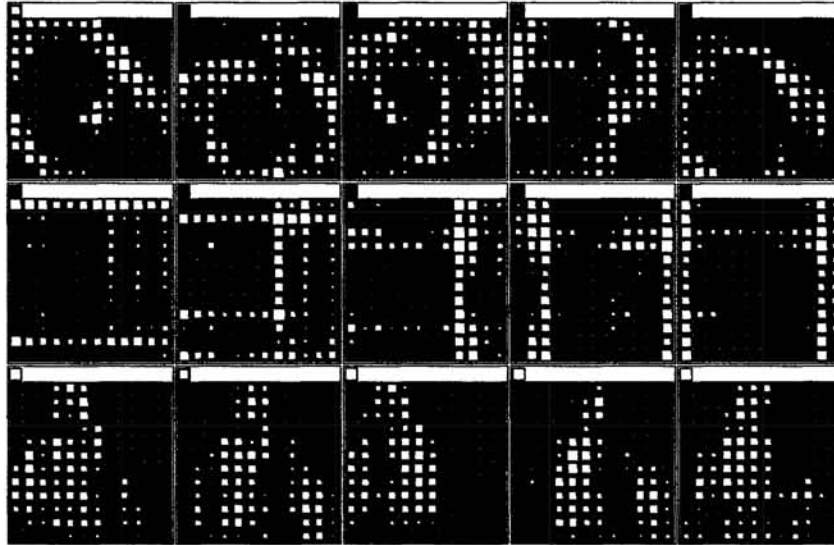

Figure 4: *Weights learned by one of the two groups of 15 units in the first layer. White weights are positive and black are negative.*

momentum to converge, but after about 1000 iterations only very small weight changes were made.

Weights learned on a typical run for one of the two groups of fifteen units are shown in Figure 4. The units' weights are displayed in three rows corresponding to units in the three blocks in the block-diagonal joint prior matrix. Units within the same block each learned different instances of the same pattern class. For example, on this run units in the first block learned to detect circles in specific positions. Units in the second block tended to learn combinations of either horizontal or vertical lines, or sometimes mixtures of the two. In the third block, units learned blurred, roughly triangular shape detectors, which for this training set were adequate to respond specifically to triangles. In all five runs the network converged to equivalent solutions (only the groups' particular shape preferences varied across runs).

Varying the number of units per block from three to five (i.e. three three-by-three blocks versus three five-by-five blocks of units) produced similar results, except that with fewer units per block, each unit tended to capture multiple instances of a particular object class in different positions.

A second layer of two groups of three units was added to the network, as shown in Figure 2 b). While keeping the first layer of weights frozen, this network was trained using exactly the same cost function as the first layer for about 30 iterations using a gradient-based learning method. This time the prior joint distribution for the two classifications was a three-by-three matrix with 80% of the density along the diagonal and 20% evenly distributed across the remainder of the distribution. Units in this layer learned to discriminate fairly well between the three object classes, as shown in Figure 5 a). On a test set with the ambiguous patterns removed (i.e., patterns containing multiple objects), units in the second layer achieved very

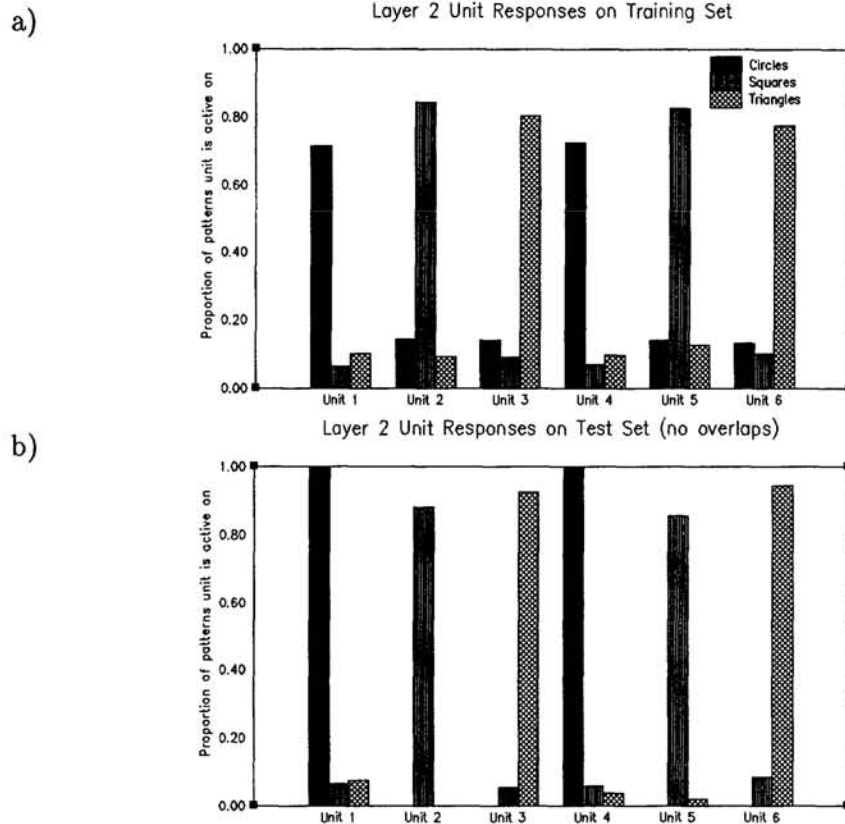

Figure 5: *Response probabilities for the six output units to each of the three shapes.*

accurate object discrimination as shown in Figure 5 b).

On ambiguous patterns containing multiple objects, the network's performance was disappointing. The output units would sometimes produce the "correct" response, i.e., all the units representing the shapes present in the image would be partially active. Most often, however, only one of the correct shapes would be detected, and occasionally the network's response indicated the wrong shape altogether. It was hoped that the diagonally dominant prior mixed with a uniform density would allow units to occasionally disagree, and they would therefore be able to represent cases of multiple objects. It may have helped to use a similar prior for the hidden layer; however, this would increase the complexity of the learning considerably.

## 3 DISCUSSION

We have shown that the algorithm can learn 2D-translation-invariant shape recognition, but it should handle equally well other types of transformations, such as rotation, scaling or even non-linear transformations. In principle, the algorithm should be applicable to real moving images; this is currently being investigated. Although we have focused here on the temporal coherence constraint, the algorithm could be applied equally well using other types of coherence, such as coherence

across space or across different sensory modalities.

Note that the units in the first layer of the network did not learn anything about the geometric transformations between translated versions of the same object; they simply learned to associate different views together. In this respect, the representation learned at the hidden layer is similar to that predicted by the "privileged views" theory of viewpoint-invariant object recognition advocated by Weinshall et al. (1990) (and others). Their algorithm learns a similar representation in a single layer of competing units with temporal Hebbian learning applied to the lateral connections between these units. However, the algorithm proposed here goes further in that it can be applied to subsequent stages of learning to discover higher-order object classes.

Yuille et al. (1994) have previously proposed an algorithm based on similar principles, which also involves maximizing the log probability of the network outputs under a prior; in one special case it is equivalent to Becker and Hinton's Imax algorithm. The algorithm proposed here differs substantially, in that we are dealing with the *ensemble-averaged* joint probabilities of two populations of units, and fitting this quantity to a prior; further, Yuille et al's scheme employs back-propagation.

One challenge for future work is to train a network with smaller receptive fields for the first layer units, on images of objects with common low-level features, such as squares and rectangles. At least three layers of weights would be required to solve this task: units in the first layer would have to learn local object parts such as corners, while units in the next layer could group parts into viewpoint-specific whole objects and in the top layer viewpoint-invariance, in principle, could be achieved.

## Acknowledgements

Helpful comments from Geoff Hinton, Peter Dayan, Tony Plate and Chris Williams are gratefully acknowledged.

## References

Becker, S. (1993). Learning to categorize objects using temporal coherence. In *Advances in Neural Information Processing Systems 5*, pages 361–368. Morgan Kaufmann.

Becker, S. and Hinton, G. E. (1992). A self-organizing neural network that discovers surfaces in random-dot stereograms. *Nature*, 355:161–163.

de Sa, V. R. (1993). Minimizing disagreement for self-supervised classification. In *Proceedings of the 1993 Connectionist Models Summer School*, pages 300–307. Lawrence Erlbaum associates.

Földiák, P. (1991). Learning invariance from transformation sequences. *Neural Computation*, 3(2):194–200.

Weinshall, D., Edelman, S., and Bülthoff, H. H. (1990). A self-organizing multiple-view representation of 3D objects. In *Advances in Neural Information Processing Systems 2*, pages 274–282. Morgan Kaufmann.

Yuille, A. L., Stelios, M. S., and Xu, L. (1994). Bayesian Self-Organization. Technical Report No. 92-10, Harvard Robotics Laboratory.